# Super-Bit Locality-Sensitive Hashing

**Jianqiu Ji**[*], **Jianmin Li**[*], **Shuicheng Yan**[†], **Bo Zhang**[*], **Qi Tian**[‡]

[*]State Key Laboratory of Intelligent Technology and Systems,
Tsinghua National Laboratory for Information Science and Technology (TNList),
Department of Computer Science and Technology,
Tsinghua University, Beijing 100084, China
`jijq10@mails.tsinghua.edu.cn,`
`{lijianmin, dcszb}@mail.tsinghua.edu.cn`
[†]Department of Electrical and Computer Engineering,
National University of Singapore, Singapore, 117576
`eleyans@nus.edu.sg`
[‡]Department of Computer Science, University of Texas at San Antonio,
One UTSA Circle, University of Texas at San Antonio, San Antonio, TX 78249-1644
`qitian@cs.utsa.edu`

## Abstract

Sign-random-projection locality-sensitive hashing (SRP-LSH) is a probabilistic dimension reduction method which provides an unbiased estimate of angular similarity, yet suffers from the large variance of its estimation. In this work, we propose the Super-Bit locality-sensitive hashing (SBLSH). It is easy to implement, which orthogonalizes the random projection vectors in batches, and it is theoretically guaranteed that SBLSH also provides an unbiased estimate of angular similarity, yet with a smaller variance when the angle to estimate is within $(0, \pi/2]$. The extensive experiments on real data well validate that given the same length of binary code, SBLSH may achieve significant mean squared error reduction in estimating pairwise angular similarity. Moreover, SBLSH shows the superiority over SRP-LSH in approximate nearest neighbor (ANN) retrieval experiments.

## 1 Introduction

Locality-sensitive hashing (LSH) method aims to hash similar data samples to the same hash code with high probability [7, 9]. There exist various kinds of LSH for approximating different distances or similarities, e.g., bit-sampling LSH [9, 7] for Hamming distance and $\ell_1$-distance, min-hash [2, 5] for Jaccard coefficient. Among them are some binary LSH schemes, which generate binary codes. Binary LSH approximates a certain distance or similarity of two data samples by computing the Hamming distance between the corresponding compact binary codes. Since computing Hamming distance involves mainly bitwise operations, it is much faster than directly computing other distances, e.g. Euclidean, cosine, which require many arithmetic operations. On the other hand, the storage is substantially reduced due to the use of compact binary codes. In large-scale applications [22, 11, 5, 17], e.g. near-duplicate image detection, object and scene recognition, etc., we are often confronted with the intensive computing of distances or similarities between samples, then binary LSH may act as a scalable solution.

### 1.1 Locality-Sensitive Hashing for Angular Similarity

For many data representations, the natural pairwise similarity is only related with the angle between the data, e.g., the normalized bag-of-words representation for documents, images, and videos, and the normalized histogram-based local features like SIFT [20]. In these cases, **angular similarity**

can serve as a similarity measurement, which is defined as $sim(a,b) = 1 - \cos^{-1}(\frac{\langle a,b \rangle}{\|a\|\|b\|})/\pi$. Here $\langle a,b \rangle$ denotes the inner product of $a$ and $b$, and $\|\cdot\|$ denotes the $\ell_2$-norm of a vector.

One popular LSH for approximating angular similarity is the sign-random-projection LSH (SRP-LSH) [3], which provides an unbiased estimate of angular similarity and is a binary LSH method. Formally, in a $d$-dimensional data space, let $v$ denote a random vector sampled from the normal distribution $\mathcal{N}(0, I_d)$, and $x$ denote a data sample, then an SRP-LSH function is defined as $h_v(x) = sgn(v^T x)$, where the sign function $sgn(\cdot)$ is defined as

$$sgn(z) = \left\{ \begin{array}{ll} 1, & z \geq 0 \\ 0, & z < 0 \end{array} \right.$$

Given two data samples $a$, $b$, let $\theta_{a,b} = \cos^{-1}(\frac{\langle a,b \rangle}{\|a\|\|b\|})$, then it can be proven that [8]

$$Pr(h_v(a) \neq h_v(b)) = \frac{\theta_{a,b}}{\pi}$$

This property well explains the essence of *locality-sensitive*, and also reveals the relation between Hamming distance and angular similarity.

By independently sampling $K$ $d$-dimensional vectors $v_1, ..., v_K$ from the normal distribution $\mathcal{N}(0, I_d)$, we may define a function $h(x) = (h_{v_1}(x), h_{v_2}(x), ..., h_{v_K}(x))$, which consists of $K$ SRP-LSH functions and thus produces $K$-bit codes. Then it is easy to prove that

$$\mathbb{E}[d_{Hamming}(h(a), h(b))] = \frac{K\theta_{a,b}}{\pi} = C\theta_{a,b}$$

That is, the expectation of the Hamming distance between the binary hash codes of two given data samples $a$ and $b$ is an unbiased estimate of their angle $\theta_{a,b}$, up to a constant scale factor $C = K/\pi$. Thus SRP-LSH provides an unbiased estimate of angular similarity.

Since $d_{Hamming}(h(a), h(b))$ follows a binomial distribution, i.e. $d_{Hamming}(h(a), h(b)) \sim \mathcal{B}(K, \frac{\theta_{a,b}}{\pi})$, its variance is $\frac{K\theta_{a,b}}{\pi}(1 - \frac{\theta_{a,b}}{\pi})$. This implies that the variance of $d_{Hamming}(h(a), h(b))/K$, i.e. $Var[d_{Hamming}(h(a), h(b))/K]$, satisfies

$$Var[d_{Hamming}(h(a), h(b))/K] = \frac{\theta_{a,b}}{K\pi}(1 - \frac{\theta_{a,b}}{\pi})$$

Though being widely used, SRP-LSH suffers from the large variance of its estimation, which leads to large estimation error. Generally we need a substantially long code to accurately approximate the angular similarity [24, 12, 23]. Since any two of the random vectors may be close to being linearly dependent, the resulting binary code may be less informative as it seems, and even contains many redundant bits. An intuitive idea would be to orthogonalize the random vectors. However, once being orthogonalized, the random vectors can no longer be viewed as independently sampled. Moreover, it remains unclear whether the resulting Hamming distance is still an unbiased estimate of the angle $\theta_{a,b}$ multiplied by a constant, and what its variance will be. Later we will give answers with theoretical justifications to these two questions.

In the next section, based on the above intuitive idea, we propose the so-called Super-Bit locality-sensitive hashing (SBLSH) method. We provide theoretical guarantees that after orthogonalizing the random projection vectors in batches, we still get an unbiased estimate of angular similarity, yet with a smaller variance when $\theta_{a,b} \in (0, \pi/2]$, and thus the resulting binary code is more informative. Experiments on real data show the effectiveness of SBLSH, which with the same length of binary code may achieve as much as 30% mean squared error (MSE) reduction compared with the SRP-LSH in estimating angular similarity on real data. Moreover, SBLSH performs best among several widely used data-independent LSH methods in approximate nearest neighbor (ANN) retrieval experiments.

## 2 Super-Bit Locality-Sensitive Hashing

The proposed SBLSH is founded on SRP-LSH. When the code length $K$ satisfies $1 < K \leq d$, where $d$ is the dimension of data space, we can orthogonalize $N$ ($1 \leq N \leq \min(K, d) = K$) of the random vectors sampled from the normal distribution $\mathcal{N}(0, I_d)$. The orthogonalization procedure

is the Gram-Schmidt process, which projects the current vector orthogonally onto the orthogonal complement of the subspace spanned by the previous vectors. After orthogonalization, these $N$ random vectors can no longer be viewed as independently sampled, thus we group their resulting bits together as an $N$-**Super-Bit**. We call $N$ the **Super-Bit depth**.

However, when the code length $K > d$, it is impossible to orthogonalize all $K$ vectors. Assume that $K = N \times L$ without loss of generality, and $1 \leq N \leq d$, then we can perform the Gram-Schmidt process to orthogonalize them in $L$ batches. Formally, $K$ random vectors $\{v_1, v_2..., v_K\}$ are independently sampled from the normal distribution $\mathcal{N}(0, I_d)$, and then divided into $L$ batches with $N$ vectors each. By performing the Gram-Schmidt process to these $L$ batches of $N$ vectors respectively, we get $K = N \times L$ projection vectors $\{w_1, w_2..., w_K\}$. This results in $K$ SBLSH functions $(h_{w_1}, h_{w_2}..., h_{w_K})$, where $h_{w_i}(x) = sgn(w_i^T x)$. These $K$ functions produce $L$ $N$-Super-Bits and altogether produce binary codes of length $K$. Figure 1 shows an example of generating 12 SBLSH projection vectors. Algorithm 1 lists the algorithm for generating SBLSH projection vectors. Note that when the Super-Bit depth $N = 1$, SBLSH becomes SRP-LSH. In other words, SRP-LSH is a special case of SBLSH. The algorithm can be easily extended to the case when the code length $K$ is not a multiple of the Super-Bit depth $N$. In fact one can even use variable Super-Bit depth $N_i$ as long as $1 \leq N_i \leq d$. With the same code length, SBLSH has the same running time $O(Kd)$ as SRP-LSH in on-line processing, i.e. generating binary codes when applying to data.

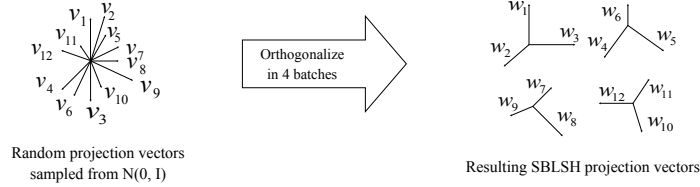

Random projection vectors
sampled from N(0, I)

Orthogonalize
in 4 batches

Resulting SBLSH projection vectors

Figure 1: An illustration of 12 SBLSH projection vectors $\{w_i\}$ generated by orthogonalizing $\{v_i\}$ in 4 batches.

---

**Algorithm 1** Generating Super-Bit Locality-Sensitive Hashing Projection Vectors

---

**Input:** Data space dimension $d$, Super-Bit depth $1 \leq N \leq d$, number of Super-Bit $L \geq 1$, resulting code length $K = N \times L$.

Generate a random matrix $H$ with each element sampled independently from the normal distribution $\mathcal{N}(0, 1)$, with each column normalized to unit length. Denote $H = [v_1, v_2, ..., v_K]$.
**for** $i = 0$ **to** $L - 1$ **do**
    **for** $j = 1$ **to** $N$ **do**
        $w_{iN+j} = v_{iN+j}$.
        **for** $k = 1$ **to** $j - 1$ **do**
            $w_{iN+j} = w_{iN+j} - w_{iN+k} w_{iN+k}^T v_{iN+j}$.
        **end for**
        $w_{iN+j} = w_{iN+j}/\|w_{iN+j}\|$.
    **end for**
**end for**
**Output:** $\tilde{H} = [w_1, w_2, ..., w_K]$.

---

## 2.1 Unbiased Estimate

In this subsection we prove that SBLSH provides an unbiased estimate of $\theta_{a,b}$ of $a, b \in \mathbb{R}^d$.

**Lemma 1.** *([8], Lemma 3.2) Let $\mathcal{S}^{d-1}$ denote the unit sphere in $\mathbb{R}^d$. Given a random vector $v$ uniformly sampled from $\mathcal{S}^{d-1}$, we have $Pr[h_v(a) \neq h_v(b)] = \theta_{a,b}/\pi$.*

**Lemma 2.** *If $v \in \mathbb{R}^d$ follows an isotropic distribution, then $\bar{v} = v/\|v\|$ is uniformly distributed on $\mathcal{S}^{d-1}$.*

This lemma can be proven by the definition of isotropic distribution, and we omit the details here.

**Lemma 3.** *Given $k$ vectors $v_1, ..., v_k \in \mathbb{R}^d$, which are sampled i.i.d. from the normal distribution $\mathcal{N}(0, I_d)$, and span a subspace $S_k$, let $P_{S_k}$ denote the orthogonal projection onto $S_k$, then $P_{S_k}$ is a random matrix uniformly distributed on the Grassmann manifold $G_{k,d-k}$.*

This lemma can be proven by applying the Theorem 2.2.1(iii), Theorem 2.2.2(iii) in [4].

**Lemma 4.** *If $P$ is a random matrix uniformly distributed on the Grassmann manifold $G_{k,d-k}$, $1 \leq k \leq d$, and $v \sim \mathcal{N}(0, I_d)$ is independent of $P$, then the random vector $\tilde{v} = Pv$ follows an isotropic distribution.*

From the uniformity of $P$ on the Grassmann manifold and the property of the normal distribution $\mathcal{N}(0, I_d)$, we can get this result directly. We give a sketch of proof below.

*Proof.* We can write $P = UU^T$, where the columns of $U = [u_1, u_2, ..., u_k]$ constitute an orthonormal basis of a random $k$-dimensional subspace. Since the standard normal distribution is 2-stable [6], $\hat{v} = U^T v = [\hat{v_1}, \hat{v_2}, ..., \hat{v_k}]^T$ is a $\mathcal{N}(0, I_k)$-distributed vector, where each $\hat{v_i} \sim \mathcal{N}(0, 1)$, and it is easy to verify that $\hat{v}$ is independent of $U$. Therefore $\tilde{v} = Pv = U\hat{v} = \Sigma_{i=1}^k \hat{v_i} u_i$. Since $u_i, ..., u_k$ can be any orthonormal basis of any $k$-dimensional subspace with equal probability density, and $\{\hat{v_1}, \hat{v_2}, ..., \hat{v_k}\}$ are i.i.d. $\mathcal{N}(0, 1)$ random variables, $\tilde{v}$ follows an isotropic distribution. $\square$

**Theorem 1.** *Given $N$ i.i.d. random vectors $v_1, v_2, ..., v_N \in \mathbb{R}^d$ sampled from the normal distribution $\mathcal{N}(0, I_d)$, where $1 \leq N \leq d$, perform the Gram-Schmidt process on them and produce $N$ orthogonalized vectors $w_1, w_2, \ldots, w_N$, then for any two data vectors $a, b \in \mathbb{R}^d$, by defining $N$ indicator random variables $X_1, X_2, ..., X_N$ as*

$$X_i = \begin{cases} 1, & h_{w_i}(a) \neq h_{w_i}(b) \\ 0, & h_{w_i}(a) = h_{w_i}(b) \end{cases}$$

*we have $\mathbb{E}[X_i] = \theta_{a,b}/\pi$, for any $1 \leq i \leq N$.*

*Proof.* Denote $S_{i-1}$ the subspace spanned by $\{w_1, ..., w_{i-1}\}$, and the orthogonal projection onto its orthogonal complement as $P_{S_{i-1}}^{\perp}$. Then $w_i = P_{S_{i-1}}^{\perp} v_i$. Denote $\bar{w} = w_i / \|w_i\|$.

For any $1 \leq i \leq N$, $\mathbb{E}[X_i] = Pr[X_i = 1] = Pr[h_{w_i}(a) \neq h_{w_i}(b)] = Pr[h_{\bar{w}}(a) \neq h_{\bar{w}}(b)]$. For $i = 1$, by Lemma 2 and Lemma 1, we have $Pr[X_1 = 1] = \theta_{a,b}/\pi$.

For any $1 < i \leq N$, consider the distribution of $w_i$. By lemma 3, $P_{S_{i-1}}$ is a random matrix uniformly distributed on the Grassmann manifold $G_{i-1,d-i+1}$, thus $P_{S_{i-1}}^{\perp} = I - P_{S_{i-1}}$ is uniformly distributed on $G_{d-i+1,i-1}$. Since $v_i \sim \mathcal{N}(0, I_d)$ is independent of $v_1, v_2, ..., v_{i-1}$, $v_i$ is independent of $P_{S_{i-1}}^{\perp}$. By Lemma 4, we have that $w_i = P_{S_{i-1}}^{\perp} v_i$ follows an isotropic distribution. By Lemma 2, $\bar{w} = w_i / \|w_i\|$ is uniformly distributed on the unit sphere in $\mathbb{R}^d$. By Lemma 1, $Pr[h_{\bar{w}}(a) \neq h_{\bar{w}}(b)] = \theta_{a,b}/\pi$. $\square$

**Corollary 1.** *For any Super-Bit depth $N$, $1 \leq N \leq d$, assuming that the code length $K = N \times L$, the Hamming distance $d_{Hamming}(h(a), h(b))$ is an **unbiased estimate** of $\theta_{a,b}$, for any two data vectors $a$ and $b \in \mathbb{R}^d$, up to a constant scale factor $C = K/\pi$.*

*Proof.* Apply Theorem 1 and we get $\mathbb{E}[d_{Hamming}(h(a), h(b))] = L \times \mathbb{E}[\Sigma_{i=1}^N X_i] = L \times \Sigma_{i=1}^N \mathbb{E}[X_i] = L \times \Sigma_{i=1}^N \theta_{a,b}/\pi = \frac{K\theta_{a,b}}{\pi} = C\theta_{a,b}$. $\square$

## 2.2 Variance

In this subsection we prove that when the angle $\theta_{a,b} \in (0, \pi/2]$, the variance of SBLSH is strictly smaller than that of SRP-LSH.

**Lemma 5.** *For the random variables $\{X_i\}$ defined in Theorem 1, we have the following equality $Pr[X_i = 1 | X_j = 1] = Pr[X_i = 1 | X_1 = 1]$, $1 \leq j < i \leq N \leq d$.*

*Proof.* $Pr[X_i = 1 | X_j = 1] = Pr[h_{w_i}(a) \neq h_{w_i}(b) | X_j = 1] = Pr[h_{v_i - \Sigma_{k=1}^{i-1} w_k w_k^T v_i}(a) \neq h_{v_i - \Sigma_{k=1}^{i-1} w_k w_k^T v_i}(b) | h_{w_j}(a) \neq h_{w_j}(b)]$. Since $\{w_1, ... w_{i-1}\}$ is a uniformly random orthonormal

basis of a random subspace uniformly distributed on Grassmann manifold, by exchanging the index $j$ and 1 we have that equals $Pr[h_{v_i - \Sigma_{k=1}^{i-1} w_k w_k^T v_i}(a) \neq h_{v_i - \Sigma_{k=1}^{i-1} w_k w_k^T v_i}(b)|h_{w_1}(a) \neq h_{w_1}(b)] = Pr[X_i = 1|X_1 = 1]$. $\quad\square$

**Lemma 6.** *For $\{X_i\}$ defined in Theorem 1, we have $Pr[X_i = 1|X_j = 1] = Pr[X_2 = 1|X_1 = 1]$, $1 \leq j < i \leq N \leq d$. Given $\theta_{a,b} \in (0, \frac{\pi}{2}]$, we have $Pr[X_2 = 1|X_1 = 1] < \frac{\theta_{a,b}}{\pi}$.*

The proof of this lemma is long, thus we provide it in the Appendix (in supplementary file).

**Theorem 2.** *Given two vectors $a, b \in \mathbb{R}^d$ and random variables $\{X_i\}$ defined as in Theorem 1, denote $p_{2,1} = Pr[X_2 = 1|X_1 = 1]$, and $S_X = \Sigma_{i=1}^N X_i$ which is the Hamming distance between the $N$-Super-Bits of $a$ and $b$, for $1 < N \leq d$, then $Var[S_X] = \frac{N\theta_{a,b}}{\pi} + N(N-1)\frac{p_{2,1}\theta_{a,b}}{\pi} - (\frac{N\theta_{a,b}}{\pi})^2$.*

*Proof.* By Lemma 6, $Pr[X_i = 1|X_j = 1] = Pr[X_2 = 1|X_1 = 1] = p_{2,1}$ when $1 \leq j < i \leq N$. Therefore $Pr[X_i = 1, X_j = 1] = Pr[X_i = 1|X_j = 1]Pr[X_j = 1] = \frac{p_{2,1}\theta_{a,b}}{\pi}$, for any $1 \leq j < i \leq N$. Therefore $Var[S_X] = \mathbb{E}[S_X^2] - \mathbb{E}[S_X]^2 = \Sigma_{i=1}^N \mathbb{E}[X_i^2] + 2\Sigma_{j<i}\mathbb{E}[X_i X_j] - N^2\mathbb{E}[X_1]^2 = \frac{N\theta_{a,b}}{\pi} + 2\Sigma_{j<i}Pr[X_i = 1, X_j = 1] - (\frac{N\theta_{a,b}}{\pi})^2 = \frac{N\theta_{a,b}}{\pi} + N(N-1)\frac{p_{2,1}\theta_{a,b}}{\pi} - (\frac{N\theta_{a,b}}{\pi})^2$. $\quad\square$

**Corollary 2.** *Denote $Var[SBLSH_{N,K}]$ as the variance of the Hamming distance produced by SBLSH, where $1 \leq N \leq d$ is the Super-Bit depth, and $K = N \times L$ is the code length. Then $Var[SBLSH_{N,K}] = L \times Var[SBLSH_{N,N}]$. Furthermore, given $\theta_{a,b} \in (0, \frac{\pi}{2}]$, if $K = N_1 \times L_1 = N_2 \times L_2$ and $1 \leq N_2 < N_1 \leq d$, then $Var[SBLSH_{N_1,K}] < Var[SBLSH_{N_2,K}]$.*

*Proof.* Since $v_1, v_2, ..., v_K$ are independently sampled, and $w_1, w_2, ..., w_K$ are produced by orthogonalizing every $N$ vectors, the Hamming distances produced by different $N$-Super-Bits are independent, thus $Var[SBLSH_{N,K}] = L \times Var[SBLSH_{N,N}]$.

Therefore $Var[SBLSH_{N_1,K}] = L_1 \times (\frac{N_1\theta_{a,b}}{\pi} + N_1(N_1-1)\frac{p_{2,1}\theta_{a,b}}{\pi} - (\frac{N_1\theta_{a,b}}{\pi})^2) = \frac{K\theta_{a,b}}{\pi} + K(N_1-1)\frac{p_{2,1}\theta_{a,b}}{\pi} - KN_1(\frac{\theta_{a,b}}{\pi})^2$. By Lemma 6, when $\theta_{a,b} \in (0, \frac{\pi}{2}]$, for $N_1 > N_2 > 1$, $0 \leq p_{2,1} < \frac{\theta_{a,b}}{\pi}$. Therefore $Var[SBLSH_{N_1,K}] - Var[SBLSH_{N_2,K}] = \frac{K\theta_{a,b}}{\pi}(N_1 - N_2)(p_{2,1} - \frac{\theta_{a,b}}{\pi}) < 0$. For $N_1 > N_2 = 1$, $Var[SBLSH_{N_1,K}] - Var[SBLSH_{N_2,K}] = \frac{K\theta_{a,b}}{\pi}(N_1 - 1)(p_{2,1} - \frac{\theta_{a,b}}{\pi}) < 0$ $\quad\square$

**Corollary 3.** *Denote $Var[SRPLSH_K]$ as the variance of the Hamming distance produced by SRP-LSH, where $K = N \times L$ is the code length and $L$ is a positive integer, $1 < N \leq d$. If $\theta_{a,b} \in (0, \frac{\pi}{2}]$, then $Var[SRPLSH_K] > Var[SBLSH_{N,K}]$.*

*Proof.* By Corollary 2, $Var[SRPLSH_K] = Var[SBLSH_{1,K}] > Var[SBLSH_{N,K}]$. $\quad\square$

#### 2.2.1 Numerical verification

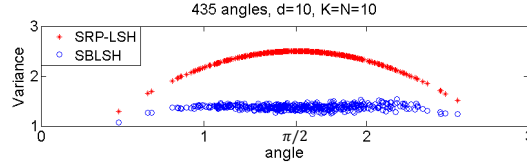

Figure 2: The variances of SRP-LSH and SBLSH against the angle $\theta_{a,b}$ to estimate.

In this subsection we verify numerically the behavior of the variances of both SRP-LSH and SBLSH for different angles $\theta_{a,b} \in (0, \pi)$. By Theorem 2, the variance of SBLSH is closely related to $p_{2,1}$ defined in Theorem 2. We randomly generate 30 points in $\mathbb{R}^{10}$, which involves 435 angles. For each angle, we numerically approximate $p_{2,1}$ using sampling method, where the sample number is 1000. We fix $K = N = d$, and plot the variances $Var[SRPLSH_N]$ and $Var[SBLSH_{N,N}]$ against various angles $\theta_{a,b}$. Figure 2 shows that when $\theta_{a,b} \in (0, \pi/2]$, SBLSH has a much smaller variance than SRP-LSH, which verifies the correctness of Corollary 3 to some extent. Furthermore, Figure 2 shows that even when $\theta_{a,b} \in (\pi/2, \pi]$, SBLSH still has a smaller variance.

## 2.3  Discussion

From Corollary 1, SBLSH provides an unbiased estimate of angular similarity. From Corollary 3, when $\theta_{a,b} \in (0, \pi/2]$, with the same length of binary code, the variance of SBLSH is strictly smaller than SRP-LSH. In real applications, many vector representations are limited in non-negative orthant with all vector entries being non-negative, e.g., bag-of-words representation of documents and images, and histogram-based representations like SIFT local descriptor [20]. Usually they are normalized to unit length, with only their orientations maintained. For this kind of data, the angle of any two different samples is limited in $(0, \pi/2]$, and thus SBLSH will provide more accurate estimation than SRP-LSH on such data. In fact, our later experiments show that even when $\theta_{a,b}$ is not constrained in $(0, \pi/2]$, SBLSH still gives much more accurate estimate of angular similarity.

## 3  Experimental Results

We conduct two sets of experiments, angular similarity estimation and approximate nearest neighbor (ANN) retrieval, to evaluate the effectiveness of our proposed SBLSH method. In the first set of experiments we directly measure the accuracy in estimating pairwise angular similarity. The second set of experiments then test the performance of SBLSH in real retrieval applications.

### 3.1  Angular Similarity Estimation

In this experiment, we evaluate the accuracy of estimating pairwise angular similarity on several datasets. Specifically, we test the effect to the estimation accuracy when the Super-Bit depth $N$ varies and the code length $K$ is fixed, and vice versa. For each preprocessed data $D$, we get $D_{LSH}$ after performing SRP-LSH, and get $D_{SBLSH}$ after performing the proposed SBLSH. We compute the angles between each pair of samples in $D$, the corresponding Hamming distances in $D_{LSH}$ and $D_{SBLSH}$. We compute the mean squared error between the true angle and the approximated angles from $D_{LSH}$ and $D_{SBLSH}$ respectively. Note that after computing the Hamming distance, we divide the result by $C = K/\pi$ and get the approximated angle.

#### 3.1.1  Datasets and Preprocessing

We conduct the experiment on the following datasets:

1) **Photo Tourism** patch dataset[1] [26], **Notre Dame**, which contains 104,106 patches, each of which is represented by a 128D SIFT descriptor (Photo Tourism SIFT); and 2) **MIR-Flickr**[2], which contains 25,000 images, each of which is represented by a 3125D bag-of-SIFT-feature histogram;

For each dataset, we further conduct a simple preprocessing step as in [12], i.e. mean-centering each data sample, so as to obtain additional mean-centered versions of the above datasets, Photo Tourism SIFT (mean), and MIR-Flickr (mean). The experiment on these mean-centered datasets will test the performance of SBLSH when the angles of data pairs are not constrained in $(0, \pi/2]$.

#### 3.1.2  The Effect of Super-Bit Depth $N$ and Code Length $K$

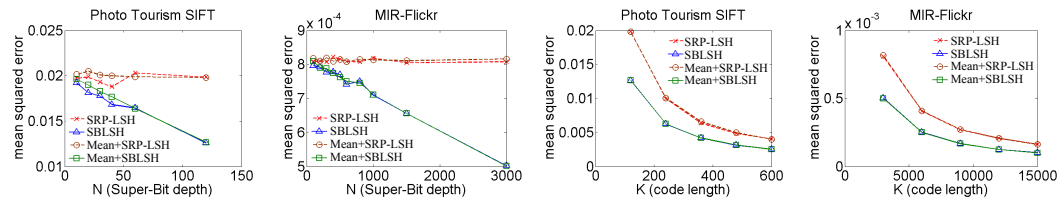

Figure 3: The effect of Super-Bit depth $N$ ($1 < N \leq \min(d, K)$) with fixed code length $K$ ($K = N \times L$), and the effect of code length $K$ with fixed Super-Bit depth $N$.

Table 1: ANN retrieval results, measured by proportion of good neighbors within query's Hamming ball of radius 3. Note that the code length $K = 30$.

| Data | E2LSH | SRP-LSH | SBLSH |
|---|---|---|---|
| Notre Dame | $0.4675 \pm 0.0900$ | $0.7500 \pm 0.0525$ | **$0.7845 \pm 0.0352$** |
| Half Dome | $0.4503 \pm 0.0712$ | $0.7137 \pm 0.0413$ | **$0.7535 \pm 0.0276$** |
| Trevi | $0.4661 \pm 0.0849$ | $0.7591 \pm 0.0464$ | **$0.7891 \pm 0.0329$** |

In each dataset, for each $(N, K)$ pair, i.e. Super-Bit depth $N$ and code length $K$, we randomly sample 10,000 data, which involve about 50,000,000 data pairs, and randomly generate SRP-LSH functions, together with SBLSH functions by orthogonalizing the generated SRP in batches. We repeat the test for 10 times, and compute the mean squared error (MSE) of the estimation.

To test the effect of Super-Bit depth $N$, we fix $K = 120$ for Photo Tourism SIFT and $K = 3000$ for MIR-Flickr respectively, and to test the effect of code length $K$, we fix $N = 120$ for Photo Tourism SIFT and $N = 3000$ for MIR-Flickr. We repeat the experiment on the mean-centered versions of these datasets, and denote the methods by Mean+SRP-LSH and Mean+SBLSH respectively.

Figure 3 shows that when using fixed code length $K$, as the Super-Bit depth $N$ gets larger $(1 < N \leq \min(d, K))$, the MSE of SBLSH gets smaller, and the gap between SBLSH and SRP-LSH gets larger. Particularly, when $N = K$, over 30% MSE reduction can be observed on all the datasets. This verifies Corollary 2 that when applying SBLSH, the best strategy would be to set the Super-Bit depth $N$ as large as possible, i.e. $\min(d, K)$. An informal explanation to this interesting phenomenon is that as the degree of orthogonality of the random projections gets higher, the code becomes more and more informative, and thus provides better estimate. On the other hand, it can be observed that the performances on the mean-centered datasets are similar as on the original datasets. This shows that even when the angle between each data pair is not constrained in $(0, \pi/2]$, SBLSH still gives much more accurate estimation.

Figure 3 also shows that with fixed Super-Bit depth $N$ SBLSH significantly outperforms SRP-LSH. When increasing the code length $K$, the accuracies of SBLSH and SRP-LSH shall both increase. The performances on the mean-centered datasets are similar as on the original datasets.

## 3.2 Approximate Nearest Neighbor Retrieval

In this subsection, we conduct ANN retrieval experiment, which compares SBLSH with two other widely used data-independent binary LSH methods: SRP-LSH and E2LSH (we use the binary version in [25, 1]). We use the datasets **Notre Dame, Half Dome** and **Trevi** from the **Photo Tourism** patch dataset [26], which is also used in [12, 10, 13] for ANN retrieval. We use 128D SIFT representation and normalize the vectors to unit norm. For each dataset, we randomly pick 1,000 samples as queries, and the rest samples (around 100,000) as the corpus for the retrieval task. We define the good neighbors to a query $q$ as the samples within the top 5% nearest neighbors (measured in Euclidean distance) to $q$. We adopt the evaluation criterion used in [12, 25], i.e. the proportion of good neighbors in returned samples that are within the query's Hamming ball of radius $r$. We set $r = 3$. Using code length $K = 30$, we repeat the experiment for 10 times and take the mean of the results. For SBLSH, we fix the Super-Bit depth $N = K = 30$. Table 1 shows that SBLSH performs best among all these data-independent hashing methods.

## 4 Relations to Other Hashing Methods

There exist different kinds of LSH methods, e.g. bit-sampling LSH [9, 7] for Hamming distance and $\ell_1$-distance, min-hash [2] for Jaccard coefficient, $p$-stable-distribution LSH [6] for $\ell_p$-distance when $p \in (0, 2]$. These data-independent methods are simple, thus easy to be integrated as a module in more complicated algorithms involving pairwise distance or similarity computation, e.g. nearest neighbor search. New data-independent methods for improving these original LSH methods have been proposed recently. [1] proposed a near-optimal LSH method for Euclidean distance. Li et al. [16] proposed b-bit minwise hash which improves the original min-hash in terms of compactness.

[17] shows that b-bit minwise hash can be integrated in linear learning algorithms for large-scale learning tasks. [14] reduces the variance of random projections by taking advantage of marginal norms, and compares the variance of SRP with regular random projections considering the margins. [15] proposed very sparse random projections for accelerating random projections and SRP.

Prior to SBLSH, SRP-LSH [3] was the only hashing method proven to provide unbiased estimate of angular similarity. The proposed SBLSH method is the first data-independent method that outperforms SRP-LSH in terms of higher accuracy in estimating angular similarity.

On the other hand, data-dependent hashing methods have been extensively studied. For example, spectral hashing [25] and anchor graph hashing [19] are data-dependent unsupervised methods. Kulis et al. [13] proposed kernelized locality-sensitive hashing (KLSH), which is based on SRP-LSH, to approximate the angular similarity in very high or even infinite dimensional space induced by any given kernel, with access to data only via kernels. There are also a bunch of works devoted to semi-supervised or supervised hashing methods [10, 21, 23, 24, 18], which try to capture not only the geometry of the original data, but also the semantic relations.

## 5    Discussion

Instead of the Gram-Schmidt process, we can use other method to orthogonalize the projection vectors, e.g. Householder transformation, which is numerically more stable. The advantage of Gram-Schmidt process is its simplicity in describing the algorithm.

In the paper we did not test the method on data of very high dimension. When the dimension is high, and $N$ is not small, the Gram-Schmidt process will be computationally expensive. In fact, when the dimension of data is very high, the random normal projection vectors $\{v_i\}_{i=1,2...,K}$ will tend to be orthogonal to each other, thus it may not be very necessary to orthogonalize the vectors deliberately.

From Corollary 2 and the results in Section 3.1.2, we can see that the closer the Super-Bit depth $N$ is to the data dimension $d$, the larger the variance reduction SBLSH achieves over SRP-LSH.

A technical report[3] (Li et al.) shows that b-bit minwise hashing almost always has a smaller variance than SRP in estimating Jaccard coefficient on binary data. The comparison of SBLSH with b-bit minwise hashing for Jaccard coefficient is left for future work.

## 6    Conclusion and Future Work

The proposed SBLSH is a data-independent hashing method which significantly outperforms SRP-LSH. We have theoretically proved that SBLSH provides an unbiased estimate of angular similarity, and has a smaller variance than SRP-LSH when the angle to estimate is in $(0, \pi/2]$. The algorithm is simple, easy to implement and can be integrated as a basic module in more complicated algorithms. Experiments show that with the same length of binary code, SBLSH achieves over 30% mean squared error reduction over SRP-LSH in estimating angular similarity, when the Super-Bit depth $N$ is close to the data dimension $d$. Moreover, SBLSH performs best among several widely used data-independent LSH methods in approximate nearest neighbor retrieval experiments. Theoretically exploring the variance of SBLSH when the angle is in $(\pi/2, \pi]$ is left for future work.

**Acknowledgments**

This work was supported by the National Basic Research Program (973 Program) of China (Grant Nos. 2013CB329403 and 2012CB316301), National Natural Science Foundation of China (Grant Nos. 91120011 and 61273023), and Tsinghua University Initiative Scientific Research Program No.20121088071, and NExT Research Center funded under the research grant WBS. R-252-300-001-490 by MDA, Singapore. And it was supported in part to Dr. Qi Tian by ARO grant W911BF-12-1-0057, NSF IIS 1052851, Faculty Research Awards by Google, FXPAL, and NEC Laboratories of America, respectively.

## Footnotes

[1] http://phototour.cs.washington.edu/patches/default.htm

[2] http://users.ecs.soton.ac.uk/jsh2/mirflickr/

[3]www.stat.cornell.edu/~li/hashing/RP_minwise.pdf

# References

[1] Alexandr Andoni and Piotr Indyk. Near-optimal hashing algorithms for approximate nearest neighbor in high dimensions. In *Annual IEEE Symposium on Foundations of Computer Science*, 2006.

[2] Andrei Z. Broder, Steven C. Glassman, Mark S. Manasse, and Geoffrey Zweig. Syntactic clustering of the web. *Computer Networks*, 29(8-13):1157–1166, 1997.

[3] Moses Charikar. Similarity estimation techniques from rounding algorithms. In *ACM Symposium on Theory of Computing*, 2002.

[4] Yasuko Chikuse. *Statistics on Special Manifolds*. Springer, February 2003.

[5] Ondrej Chum, James Philbin, and Andrew Zisserman. Near duplicate image detection: min-hash and tf-idf weighting. In *British Machine Vision Conference*, 2008.

[6] Mayur Datar, Nicole Immorlica, Piotr Indyk, and Vahab S. Mirrokni. Locality-sensitive hashing scheme based on p-stable distributions. In *Symposium on Computational Geometry*, 2004.

[7] Aristides Gionis, Piotr Indyk, and Rajeev Motwani. Similarity search in high dimensions via hashing. In *International Conference on Very Large Databases*, 1999.

[8] Michel X. Goemans and David P. Williamson. Improved approximation algorithms for maximum cut and satisfiability problems using semidefinite programming. *Journal of the ACM*, 42(6):1115–1145, 1995.

[9] Piotr Indyk and Rajeev Motwani. Approximate nearest neighbors: Towards removing the curse of dimensionality. In *ACM Symposium on Theory of Computing*, 1998.

[10] Prateek Jain, Brian Kulis, and Kristen Grauman. Fast image search for learned metrics. In *IEEE Conference on Computer Vision and Pattern Recognition*, 2008.

[11] Hervé Jégou, Matthijs Douze, and Cordelia Schmid. Product quantization for nearest neighbor search. *IEEE Trans. Pattern Anal. Mach. Intell.*, 33(1):117–128, 2011.

[12] Brian Kulis and Trevor Darrell. Learning to hash with binary reconstructive embeddings. In *Advances in Neural Information Processing Systems*, 2009.

[13] Brian Kulis and Kristen Grauman. Kernelized locality-sensitive hashing for scalable image search. In *IEEE International Conference on Computer Vision*, 2009.

[14] Ping Li, Trevor Hastie, and Kenneth Ward Church. Improving random projections using marginal information. In *COLT*, pages 635–649, 2006.

[15] Ping Li, Trevor Hastie, and Kenneth Ward Church. Very sparse random projections. In *KDD*, pages 287–296, 2006.

[16] Ping Li and Arnd Christian König. b-bit minwise hashing. In *International World Wide Web Conference*, 2010.

[17] Ping Li, Anshumali Shrivastava, Joshua L. Moore, and Arnd Christian König. Hashing algorithms for large-scale learning. In *Advances in Neural Information Processing Systems*, 2011.

[18] Wei Liu, Jun Wang, Rongrong Ji, Yu-Gang Jiang, and Shih-Fu Chang. Supervised hashing with kernels. In *CVPR*, pages 2074–2081, 2012.

[19] Wei Liu, Jun Wang, Sanjiv Kumar, and Shih-Fu Chang. Hashing with graphs. In *ICML*, pages 1–8, 2011.

[20] David G. Lowe. Distinctive image features from scale-invariant keypoints. *International Journal of Computer Vision*, 60(2):91–110, 2004.

[21] Yadong Mu, Jialie Shen, and Shuicheng Yan. Weakly-supervised hashing in kernel space. In *IEEE Conference on Computer Vision and Pattern Recognition*, 2010.

[22] Antonio Torralba, Robert Fergus, and William T. Freeman. 80 million tiny images: A large data set for nonparametric object and scene recognition. *IEEE Trans. Pattern Anal. Mach. Intell.*, 30(11):1958–1970, 2008.

[23] Jun Wang, Sanjiv Kumar, and Shih-Fu Chang. Sequential projection learning for hashing with compact codes. In *International Conference on Machine Learning*, 2010.

[24] Jun Wang, Sanjiv Kumar, and Shih-Fu Chang. Semi-supervised hashing for large scale search. *IEEE Transactions on Pattern Analysis and Machine Intelligence*, 99(PrePrints), 2012.

[25] Yair Weiss, Antonio Torralba, and Robert Fergus. Spectral hashing. In *Advances in Neural Information Processing Systems*, 2008.

[26] Simon A. J. Winder and Matthew Brown. Learning local image descriptors. In *IEEE Conference on Computer Vision and Pattern Recognition*, 2007.

